# Fast Network Pruning and Feature Extraction Using the Unit-OBS Algorithm

**Achim Stahlberger and Martin Riedmiller**
Institut für Logik, Komplexität und Deduktionssysteme
Universität Karlsruhe, 76128 Karlsruhe, Germany
email: stahlb@ira.uka.de, riedml@ira.uka.de

## Abstract

The algorithm described in this article is based on the OBS algorithm by Hassibi, Stork and Wolff ([1] and [2]). The main disadvantage of OBS is its high complexity. OBS needs to calculate the inverse Hessian to delete only one weight (thus needing much time to prune a big net). A better algorithm should use this matrix to remove more than only one weight, because calculating the inverse Hessian takes the most time in the OBS algorithm.

The algorithm, called Unit–OBS, described in this article is a method to overcome this disadvantage. This algorithm only needs to calculate the inverse Hessian once to remove one whole unit thus drastically reducing the time to prune big nets.

A further advantage of Unit–OBS is that it can be used to do a feature extraction on the input data. This can be helpful on the understanding of unknown problems.

## 1 Introduction

This article is based on the technical report [3] about speeding up the OBS algorithm. The main target of this work was to reduce the high complexity $O(n^2 p)$ of the OBS algorithm in order to use it for big nets in a reasonable time. Two "exact" algorithms were developed which lead to exactly the same results as OBS but using less time. The first with time $O(n^{1.8} p)$ makes use of Strassens' fast matrix multiplication algorithm. The second algorithm uses algebraic transformations to speed up calculation and needs time $O(np^2)$. This algorithm is faster than OBS in the special case of $p < n$.

To get a much higher speedup than these exact algorithms can do, an improved OBS algorithm was developed which reduces the runtime needed to prune a big network drastically. The basic idea is to use the inverse Hessian to remove a group of weights instead of only one, because the calculation of this matrix takes the most time in the OBS algorithm. This idea leads to an algorithm called Unit–OBS that is able to remove whole units.

Unit–OBS has two main advantages: First it is a fast algorithm to prune big nets, because whole units are removed in every step instead of slow pruning weight by weight. On the other side it can be used to do a feature extraction on the input data by removing unimportant input units. This is helpful for the understanding of unknown problems.

## 2   Optimal Brain Surgeon

This section gives a small summary of the OBS algorithm described by Hassibi, Stork and Wolff in [1] and [2]. As they showed the increase in error (when changing weights by $\Delta w$) is

$$\Delta E = \frac{1}{2}\Delta w^T H \Delta w \tag{1}$$

where $H$ is the Hessian matrix. The goal is to eliminate weight $w_q$ and minimize the increase in error given by Eq. 1. Eliminating $w_q$ can be expressed by $w_q + \Delta w_q = 0$ which is equivalent to $(w+\Delta w)^T e_q = 0$ where $e_q$ is the unit vector corresponding to weight $w_q$ ($w^T e_q = w_q$). Solving this extremum problem with side condition using Lagrange's method leads to the solution

$$\Delta E = \frac{w_q{}^2}{2H^{-1}{}_{qq}} \tag{2}$$

$$\Delta w = -\frac{w_q}{H^{-1}{}_{qq}}H^{-1}e_q \tag{3}$$

$H^{-1}{}_{qq}$ denotes the element $(q,q)$ of matrix $H^{-1}$. For every weight $w_q$ the minimal increase in error $\Delta E(w_q)$ is calculated and the weight which leads to overall minimum will be removed and all other weights be adapted referring to Eq. 3. Hassibi, Stork and Wolff also showed how to calculate $H^{-1}$ using time $O(n^2p)$ where $n$ is the number of weights and $p$ the number of pattern.

The main disadvantage of the OBS algorithm is that it needs time $O(n^2p)$ to remove only one weight thus needing much time to prune big nets. The basic idea to soften this disadvantage is to use $H^{-1}$ to remove more than only one weight! This generalized OBS algorithm is described in the next section.

## 3   Generalized OBS (G–OBS)

This section shows a generalized OBS algorithm (G–OBS) which can be used to delete $m$ weights in one step with minimal increase in error. Like in the OBS algorithm the increase in error is given by $\Delta E = \frac{1}{2}\Delta w^T H \Delta w$. But the condition $w_q + \Delta w_q = 0$ is replaced by the generalized condition

$$(w + \Delta w)^T M = 0 \quad \text{with} \quad M = (e_{q_1}\, e_{q_2}\, \dots\, e_{q_m}) \tag{4}$$

where $M$ is the selection matrix (selecting the weights to be removed) and $q_1, q_2, \ldots, q_m$ are the indices of the weights that will be removed. Solving this extremum problem with side condition using Lagrange's method leads to the solution

$$\Delta E = \frac{1}{2} w^T M (M^T H^{-1} M)^{-1} M^T w \tag{5}$$

$$\Delta w = -H^{-1} M (M^T H^{-1} M)^{-1} M^T w \tag{6}$$

Choosing $M = e_q$ Eq. 5 and 6 reduce to Eq. 2 and 3. This shows that OBS is (as expected) a special case of G–OBS. The problem of calculating $H^{-1}$ was already solved by Hassibi, Stork and Wolff ([1] and [2]).

## 4   Analysis of G–OBS

Hassibi, Stork and Wolff ([1] and [2]) showed that the time to calculate $H^{-1}$ is in $O(n^2 p)$. The calculation of $\Delta E$ referring to Eq. 5 needs time $O(m^3)$[†] where $m$ is the number of weights to be removed. The calculation of $\Delta w$ (Eq. 6) needs time $O(nm + m^3)$.

The problem within this solution consists of not knowing which weights should be deleted and thus $\Delta E$ has to be calculated for *all* possible combinations to find the global minimum in error increase. Choosing $m$ weights out of $n$ can be done with $\binom{n}{m}$ possible combinations. This takes time $\binom{n}{m} O(m^3)$ to find the minimum. Therefore the total runtime of the generalized OBS algorithm to remove $m$ weights (with minimal increase in error) is

$$T_{\text{G–OBS}} = O(n^2 p + \binom{n}{m} m^3).$$

The problem is that for $m > 3$ the term $\binom{n}{m} m^3$ dominates and $T_{\text{G–OBS}}$ is in $\Omega(n^4)$. In other words G–OBS can be used only to remove a maximum of three weights in one step. But this means little advantage over OBS.

To overcome this problem the set of possible combinations has to be restricted to a small subset of combinations that seem to be "good" combinations. This reduces the term $\binom{n}{m} m^3$ to a reasonable amount. One way to do this is that a good combination exists of all outgoing connections of a unit. This reduces the number of combinations to the number of units! The basic idea for that subset is: If all outgoing connections of a unit can be removed then the whole unit can be deleted because it can not influence the net output anymore. Therefore choosing this subset leads to an algorithm called Unit–OBS that is able to remove whole units without the need to recalculate $H^{-1}$.

## 5   Special Case of G–OBS: Unit–OBS

With the results of the last sections we can now describe an algorithm called Unit–OBS to remove whole units.

  1. Train a network to minimum error.

---

[†]$M$ is a matrix of special type and thus the calculation of $(M^T H^{-1} M)$ needs only $O(m^2)$ operations!

2. Compute $H^{-1}$.

3. For each unit $u$

   (a) Compute the indices $q_1, q_2, \ldots, q_{m(u)}$ of the outgoing connections of unit $u$ where $m(u)$ is the number of outgoing connections of unit $u$.

   (b) $M := (e_{q_1} \, e_{q_2} \, \ldots \, e_{q_{m(u)}})$

   (c) $\Delta E(u) := \frac{1}{2} w^T M (M^T H^{-1} M)^{-1} M^T w$

4. Find the $u_0$ that gives the smallest increase in error $\Delta E(u_0)$.

5. $M := M(u_0)$ (refer to steps 3.(a) and 3.(b))

6. $\Delta w := -H^{-1} M (M^T H^{-1} M)^{-1} M^T w$

7. Remove unit $u_0$ and use $\Delta w$ to update all weights.

8. Repeat steps 2 to 7 until a break criteria is reached.

Following the analysis of G–OBS the time to remove one unit is

$$T_{\text{Unit-OBS}} = O(n^2 p + u m^3) \tag{7}$$

where $u$ is the number of units in the network and $m$ is the maximum number of outgoing connections. If $m$ is much smaller than $n$ we can neglect the term $u m^3$ and the main problem is to calculate $H^{-1}$. Therefore, if $m$ is small, we can say that Unit–OBS needs the same time to remove a whole unit as OBS needs to remove a single weight. The speedup when removing units with an average of $s$ outgoing connections should then be $s$.

## 6    Simulation results

### 6.1    The Monk–1 benchmark

Unit–OBS was applied to the MONK's problems because the underlying logical rules are well known and it is easy to say which input units are important to the problem and which input units can be removed. The simulations showed that in no case Unit–OBS removed a wrong unit and that it has the ability to remove all unimportant input units.

Figure 1 shows a MONK–1–net pruned with Unit–OBS. This net is the minimal network that can be found by Unit–OBS. Table 1 shows the speedup of Unit-OBS compared to OBS to find an equal–size network for the MONK–1 problem.

The network shown in Fig. 1 is only minimal in the number of units but not minimal with respect to the number of weights. Hassibi, Stork and Wolff ([1] and [2]) found a network with only 14 weights by applying OBS (Fig. 3). In the framework of Unit–OBS, OBS can be used to do further pruning on the network after all possible units have been pruned. The advantage lies in the fact that now the time consuming OBS–algorithm is applied to a much smaller network (22 weights instead of 58). The result of this combination of Unit–OBS and OBS is a network with only 14 weights (Fig. 2) which has also 100 % accuracy like the minimal net found by OBS (see Table 1).

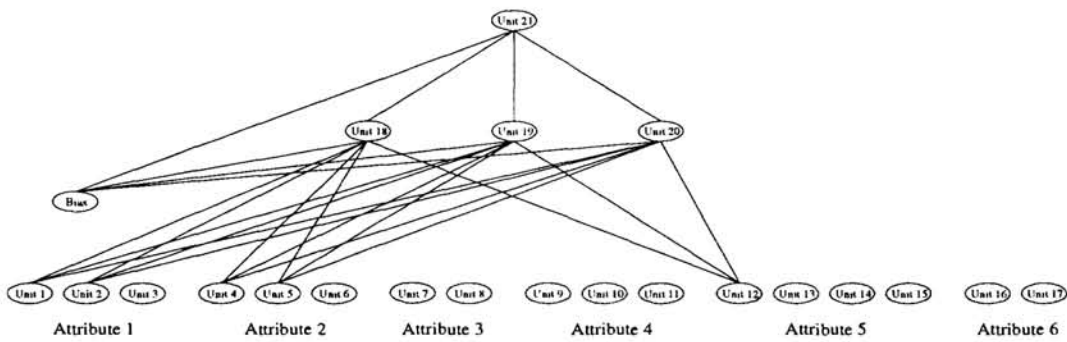

Figure 1: MONK–1–net pruned with Unit–OBS, 22 weights. All unimportant units are removed and this net needs less units than the minimal network found by OBS!

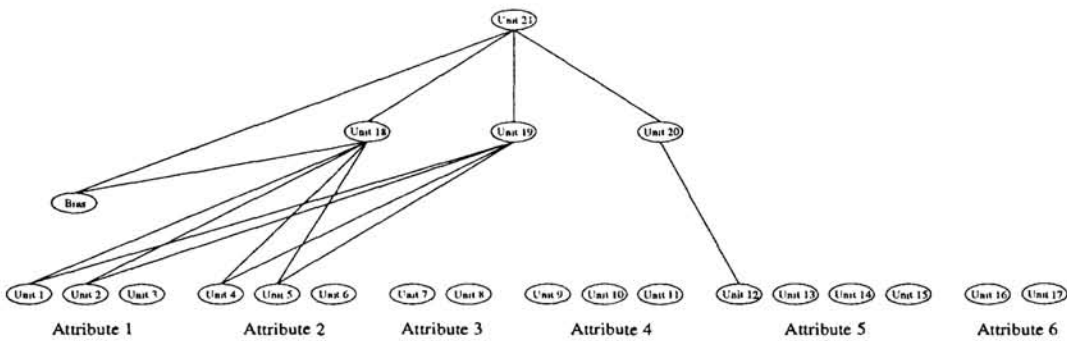

Figure 2: Minimal network (14 weights) for the MONK–1 problem found by the combination of Unit–OBS with OBS. The logical rule for the MONK–1 problem is more evident in this network than in the minimal network found by OBS (comp. Fig. 3).

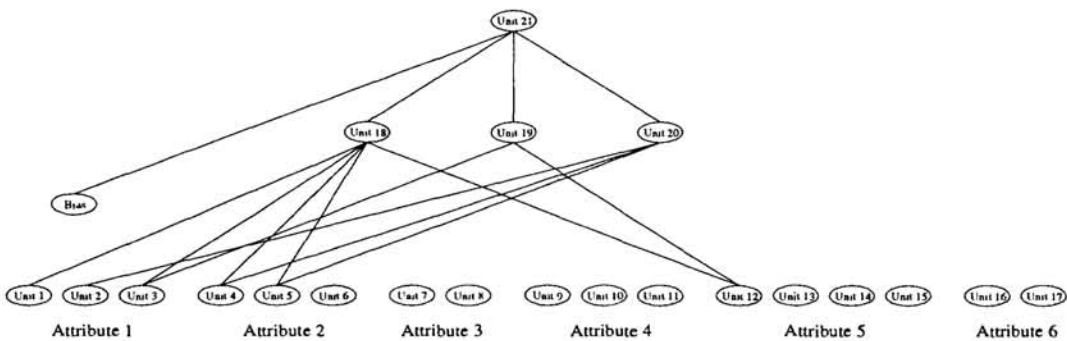

Figure 3: Minimal network (14 weights) for the MONK–1 problem found by OBS (see [1] and [2]).

| algorithm | # weights | topology | speedup[‡] | perf. train | perf. test |
|---|---|---|---|---|---|
| no pruning | 58 | 17-3-1 | - | 100 % | 100 % |
| OBS | 14 | **6**-3-1 | 1.0 | 100 % | 100 % |
| Unit-OBS | 22 | **5**-3-1 | 2.8 | 100 % | 100 % |
| Unit-OBS + OBS | 14 | **5**-3-1 | 2.6 | 100 % | 100 % |

Table 1: The Monk–1 problem

For the initial Monk–1 network the maximum number of outgoing connections ($m$ in Eq. 7) is 3 and this is much smaller than the number of weights. The average number of outgoing connections of the removed units is 3 and therefore we expect a speedup by factor 3 (compare Table 1).

By comparing the two minimal nets found by Unit–OBS/OBS (Fig. 2) and OBS (Fig. 3) it can be seen that the underlying logical rule (out=1 ⇔ Attribut_1=Attribut_2 or Attribut_5=1) is more evident in the network found by Unit–OBS/OBS. The other advantage of Unit–OBS is that it needs only 38 % of the time OBS needs to find this minimal network. This advantage makes it possible to apply Unit–OBS to big nets for which OBS is not useful because of its long computation time.

## 6.2   The Thyroid Benchmark

The following describes the application of pruning on a medical classification problem. The task is to classify measured data values of patients into three categories. The output of the three layered feedforward network therefore consists of three neurons indicating the corresponding class. The input consists of 21 both continuos and binary signals.

The task was first described in [4]. The results obtained there are shown in the first row of Table 2. The initially used network has 21 input neurons, 10 hidden and 3 output neurons, which are fully connected using shortcut connections.

When applying OBS to prune the network weights, more than 90 % of the weights can be pruned. However, over 8 hours of cpu-time on a sparc workstation are used to do so (row 2 in Table 2). The solution finally found by OBS uses only 8 of the originally 21 input features. The pruned network shows a slightly improved classification rate on the test set.

Unit–OBS finds a solution with 41 weights in only 76 minutes of cpu-time. In comparison to the original OBS algorithm, Unit–OBS is about 8 times as fast when deleting the same number of weights. Also another important fact can be seen from the result: The Unit–OBS network considers only 7 of the originally 21 inputs, 1 less than the weight-focused OBS–algorithm. The number of hidden units is reduced to 2 units, 5 units less than the OBS network uses.

When further looking for an absolute minimum in the number of used weights, the Unit–OBS network can be additionally pruned using OBS. This finally leeds to an optimized network with only 24 weights. The classification performance of this very

---

[‡]Compared to OBS deleting the same number of weights.

small network is 98.5 % which is even slightly better than obtained by the much bigger initial net.

| algorithm | # weights | topology | speedup | cpu-time | perf. test |
|:---:|:---:|:---:|:---:|:---:|:---:|
| no pruning | 316 | 21-10-3 | - | - | 98.4% |
| OBS | 28 | **8-7-3** | 1.0 | 511 min. | 98.5% |
| Unit-OBS | 41 | **7-2-3** | 7.8 | 76 min. | 98.4% |
| Unit-OBS + OBS | 24 | **7-2-3** | - | 137 min. | 98.5% |

Table 2: The thyroid benchmark

## 7   Conclusion

The article describes an improvement of the OBS–algorithm introduced in [1] called Generalized OBS (G–OBS). The underlying idea is to exploit second order information to delete *mutliple* weights at once. The aim to reduce the number of different weight groups leads to the formulation of the Unit-OBS algorithm, which considers the outgoing weights of one unit as a group of candidate weights: When all the weights of a unit can be deleted, the unit itself can be pruned. The new Unit-OBS algorithm has two major advantages: First, it considerably accelerates pruning by a speedup factor which lies in the range of the average number of outgoing weights of each unit. Second, deleting complete units is especially interesting to determine the input features which *really* contribute to the computation of the output. This information can be used to get more insight in the underlying problem structure, e.g. to facilitate the process of rule extraction.

## References

[1] B. Hassibi, D. G. Storck: *Second Order Derivatives for Network Pruning: Optimal Brain Surgeon*. Advances in Neural Information Processing Systems 5, Morgan Kaufmann, 1993, pages 164–171.

[2] B. Hassibi, D. G. Stork, G. J. Wolff: *Optimal Brain Surgeon and general Network Pruning*. IEEE International Conference on Neural Networks, 1993 Volume 1, pages 293–299.

[3] A. Stahlberger: *OBS – Verbesserungen und neue Ansätze*. Diplomarbeit, Universität Karlsruhe, Institut für Logik, Komplexität und Deduktionssysteme, 1996.

[4] W. Schiffmann, M. Joost, R. Werner: *Optimization of the Backpropagation Algorithm for Training Multilayer Perceptrons*. Technical Report, University of Koblenz, Institute of Physics, 1993.